# Some results on convergent unlearning algorithm

**Serguei A. Semenov & Irina B. Shuvalova**
Institute of Physics and Technology
Prechistenka St. 13/7
Moscow 119034, Russia

## Abstract

In this paper we consider probabilities of different asymptotics of convergent unlearning algorithm for the Hopfield-type neural network (Plakhov & Semenov, 1994) treating the case of unbiased random patterns. We show also that failed unlearning results in total memory breakdown.

## 1  INTRODUCTION

In the past years the unsupervised learning schemes arose strong interest among researchers but for the time being a little is known about underlying learning mechanisms, as well as still less rigorous results like convergence theorems were obtained in this field. One of promising concepts along this line is so called "unlearning" for the Hopfield-type neural networks (Hopfield et al, 1983, van Hemmen & Klemmer, 1992, Wimbauer et al, 1994). Elaborating that elegant ideas the convergent unlearning algorithm has recently been proposed (Plakhov & Semenov, 1994), executing without patterns presentation. It is aimed at to correct initial Hebbian connectivity in order to provide extensive storage of arbitrary correlated data.

This algorithm is stated as follows. Pick up at iteration step $m$, $m = 0, 1, 2, \ldots$ a random network state $S^{(m)} = (S_1^{(m)}, \ldots, S_N^{(m)})$, with the values $S_i^{(m)} = \pm 1$ having equal probability $1/2$, calculate local fields generated by $S^{(m)}$

$$h_i^{(m)} = \sum_{i=1}^{N} J_{ij}^{(m)} S_j^{(m)}, \quad i = 1, \ldots, N,$$

and then update the synaptic weights by

$$J_{ij}^{(m+1)} = J_{ij}^{(m)} - \varepsilon N^{-1} h_i^{(m)} h_j^{(m)}, \quad i, j = 1, \ldots, N. \tag{1}$$

Here $\varepsilon > 0$ stands for the unlearning strength parameter. We stress that self-interactions, $J_{ii}$, are necessarily involved in the iteration process. The initial condition for (1) is given by the Hebb matrix, $J_{ij}^{(0)} = J_{ij}^H$:

$$J_{ij}^H = N^{-1} \sum_{\mu=1}^{p} \xi_i^\mu \xi_j^\mu \qquad (2)$$

with arbitrary $(\pm 1)$-patterns $\xi^\mu$, $\mu = 1, \ldots, p$.

For $\varepsilon < \varepsilon_c$, the (rescaled) synaptic matrix has been proven to converge with probability one to the projection one on the linear subspace spanned by maximal subset of linearly independent patterns (Plakhov & Semenov, 1994). As the sufficient condition for that convergence to occur, the value of unlearning strength $\varepsilon$ should be less than $\varepsilon_c = \lambda_{\max}^{-1}$ where $\lambda_{\max}$ denotes the largest eigenvalue of the Hebb matrix. Very often in real-world situations there are no means to know $\varepsilon_c$ in advance, and therefore it is of interest to explore asymptotic behaviour of iterated synaptic matrix for arbitrary values of $\varepsilon$. As it is seen, there are only three possible limiting behaviours of the normalized synaptic matrix (Plakhov 1995, Plakhov & Semenov, 1995). The corresponding convergence theorems relate corresponding spectrum dynamics to limiting behaviour of normalized synaptic matrix $\tilde{J} = J/\|J\|$ ( $\|J\| = (\sum_{i,j=1}^{N} J_{ij}^2)^{1/2}$ ) which can be described in terms of $\lambda_{\min}^{(m)}$ the smallest eigenvalues of $J^{(m)}$:

I. if $\lambda_{\min}^{(m)} = 0$ for every $m = 0, 1, 2, \ldots$, with multiplicity of zero eigenvalue being fixed, then

$$(A) \qquad \lim_{m \to \infty} \tilde{J}_{ij}^{(m)} = s^{-1/2} P_{ij}$$

where $P$ marks the projection matrix on the linear subspace $\mathcal{L} \subset \mathbf{R}^N$ spanned by the nominated patterns set $\xi^\mu$, $\mu = 1, \ldots, p$, $s = \dim \mathcal{L} \leq p$;

II. if $\lambda_{\min}^{(m)} = 0$, $m = 0, 1, 2, \ldots$, besides at some (at least one) steps multiplicity of zero eigenvalue increases, then

$$(B) \qquad \lim_{m \to \infty} \tilde{J}_{ij}^{(m)} = s'^{-1/2} P_{ij}'$$

where $P'$ is the projector on some subspace $\mathcal{L}' \subset \mathcal{L}$, $s' = \dim \mathcal{L}' < s$;

III. if $\lambda_{\min}^{(m)} < 0$ starting from some value of $m$, then

$$(C) \qquad \lim_{m \to \infty} \tilde{J}_{ij}^{(m)} = -\xi_i \xi_j \qquad (3)$$

with some (not a $\pm 1$) unity random vector $\xi = (\xi_1, \ldots, \xi_N)$.

These three cases exhaust all possible asymptotic behaviours of $\tilde{J}_{ij}^{(m)}$, that is their total probability is unity: $P_A + P_B + P_C = 1$. The patterns set is supposed to be fixed.

The convergence theorems say nothing about relative probabilities to have specific asymptotics depending on model parameters. In this paper we present some general results elucidating this question and verify them by numerical simulation.

We show further that the limiting synaptic matrix for the case $(C)$ which is the projector on $-\xi \in \mathcal{L}$ cannot maintain any associative memory. Brief discussion on the retrieval properties of the intermediate case $(B)$ is also given.

## 2  PROBABILITIES OF POSSIBLE LIMITING BEHAVIOURS OF $\tilde{J}^{(m)}$

The unlearning procedure under consideration is stochastic in nature. Which result of iteration process, $(A), (B)$ or $(C)$, will realize depends upon the value of $\varepsilon$, size and statistical properties of the patterns set $\{\xi^\mu, \ \mu = 1, \dots, p\}$, and realization of unlearning sequence $\{S^{(m)}, \ m = 0, 1, 2, \dots\}$.

Under fixed patterns set probabilities of appearance of each limiting behaviour of synaptic matrix is determined by the value of unlearning strength $\varepsilon$ only. In this section we consider these probabilities as a function of $\varepsilon$.

Generally speaking, considered probabilities exhibit strong dependence on patterns set, making impossible to calculate them explicitly. It is possible however to obtain some general knowledge concerning that probabilities, namely: $P_A(\varepsilon) \to 1$ as $\varepsilon \to 0+$, and hence, $P_{B,C}(\varepsilon) \to 0$, otherwise $P_C(\varepsilon) \to 1$ as $\varepsilon \to \infty$, and $P_{A,B}(\varepsilon) \to 0$, because of $P_A + P_B + P_C = 1$. This means that the risk to have failed unlearning rises when $\varepsilon$ increases. Specifically, we are able to prove the following:

**Proposition.**  *There exist positive $\varepsilon_1$ and $\varepsilon_2$ such that $P_A(\varepsilon) = 1, \quad 0 < \varepsilon < \varepsilon_1$, and $P_C(\varepsilon) = 1, \quad \varepsilon_2 < \varepsilon$.*

Before passing to the proof we bring forward an alternative formulation of the above stated classification. After multiplying both sides of (1) by $S_i^{(m)} S_j^{(m)}$ and summing up over all $i$ and $j$, we obtain in the matrix notation

$$S^{(m)T} J^{(m+1)} S^{(m)} = \Delta_m S^{(m)T} J^{(m)} S^{(m)} \tag{4}$$

where the contraction factor $\Delta_m = 1 - \varepsilon N^{-1} S^{(m)T} J^{(m)} S^{(m)}$ controls the asymptotics of $\tilde{J}^{(m)}$, as it is suggested by detailed analysis (Plakhov & Semenov, 1995). (Here and below superscript $T$ designates the transpose.) The hypothesis of convergence theorems can be thus restated in terms of $\Delta_m$, instead of $\lambda_{\min}^{(m)}$, respectively:
I. $\Delta_m > 0 \ \forall m$;  II. $\Delta_m = 0$ for $l$ steps $m_1, \dots, m_l$;  III. $\Delta_m < 0$ at some step $m$.

*Proof.*  It is obvious that $\Delta_m \geq 1 - \varepsilon \lambda_{\max}^{(m)}$ where $\lambda_{\max}^{(m)}$ marks the largest eigenvalue of $J^{(m)}$. From (4), it follows that the sequence $\{\lambda_{\max}^{(m)}, \ m = 0, 1, 2, \dots\}$ is nonincreasing, and consequently $\Delta_m \geq 1 - \varepsilon \lambda_{\max}^{(0)}$ with

$$\lambda_{\max}^{(0)} = \sup_{|x|=1} x^T J^H x = \sup_{|x|=1} N^{-1} \sum_{\mu=1}^{p} \left( \sum_i \xi_i^\mu x_i \right)^2$$

$$\leq \sup_{|x|=1} N^{-1} \sum_{\mu=1}^{p} \sum_{i=1}^{N} (\xi_i^\mu)^2 \sum_{i=1}^{N} x_i^2 = p.$$

From this, it is straightforward to see that, if $\varepsilon < p^{-1}$, then $\Delta_m > 0$ for any $m$. By convergence theorem (Plakhov & Semenov, 1995) iteration process (1) thus leads to the limiting relation $(A)$.

Let by definition $\gamma = \min_S N^{-1} S^T J^H S$ where minimum is taken over such $(\pm 1)$-vectors $S$ for which $J^H S \neq 0$ ($\gamma > 0$, in view of positive semidefiniteness of $J^H$), and put $\varepsilon > \gamma^{-1}$. Let us further denote by $n$ the iteration step such that $J^H S^{(m)} = 0, \ m = 0, 1, \dots, n-1$ and $J^H S^{(n)} \neq 0$. Needless to say that this condition may be satisfied even for the initial step $n = 0$: $J^H S^{(0)} \neq 0$. At step $n$ one has

$$\Delta_n = 1 - \varepsilon N^{-1} S^{(n)T} J^H S^{(n)} \leq 1 - \varepsilon \gamma < 0.$$

The latter implies loss of positive semidefiniteness of $J^{(m)}$, what results in asymptotics $(C)$ (Plakhov, 1995, Plakhov & Semenov, 1995). By choosing $\varepsilon_1 = p^{-1}$ and $\varepsilon_2 = \gamma^{-1}$ we come to the statement of Proposition.

Comparison of numerical estimates of considered probabilities with analytical approximations can be done on simple patterns statistics. In what follows the patterns are assumed to be random and unbiased.

The dependence $P(\varepsilon)$ has been found in computer simulation with unbiased random patterns. It is worth noting, by passing, that calculation $\Delta_m$ using current simulation data supplies a good control of unlearning process owing to an alternative formulation of convergence theorems. In simulation we calculate $P_A^N(\varepsilon)$ averaged over the sets of unbiased random patterns, as well as over the realizations of unlearning sequence. As $N$ increases, with $\alpha = p/N$ remaining fixed, the curves slope steeply down approaching step function $P_A^\infty(\varepsilon) = \theta(\varepsilon - \alpha^{-1})$ (Plakhov & Semenov, 1995). Without presenting of derivation or proof we will advance the reasoning suggestive of it. First it can be checked that $\Delta_m$ is a selfaveraging quantity with mean $1 - \varepsilon N^{-1} \mathrm{Tr} J^{(m)}$ and variance vanishing as $N$ goes to infinity. Initially one has $N^{-1} \mathrm{Tr} J^H = \alpha$, and obviously the sequence $\{ \mathrm{Tr} J^{(m)}, \ m = 0, 1, 2, \ldots \}$ is nonincreasing. Therefore $\Delta_0 = 1 - \varepsilon \alpha$, and all others $\Delta_m$ are not less than $\Delta_0$. If one chooses $\varepsilon < \alpha^{-1}$, then all $\Delta_m$ will be positive, and the case $(A)$ will realize. On the other hand, when $\varepsilon > \alpha^{-1}$, we have $\Delta_0 < 0$, and the case $(C)$ will take place.

What is probability for asymptotics $(B)$ to appear? We will adduce an argument (detailed analysis (Plakhov & Semenov, 1995) is rather cumbersome and omitted here) indicating that this probability is quite small. First note that given patterns set it is nonzero for isolated values of $\varepsilon$ only. Under the assumption that the patterns are random and unbiased, we have calculated probability of $l$-fold appearance $\Delta_m = 0$ summed up over that isolated values of $\varepsilon$. Using Gaussian approximation at large $N$, we have found that probability scales with $N$ as $N^{l/2+2-2^{l+m+1}}$. The total probability can then be obtained through summing up over integer values $l: \ 0 < l < s$ and all the iteration steps $m = 0, 1, 2, \ldots$. As a result, the main contribution to the total probability comes from $m = 0$ term which is of the order $N^{-3/2}$.

## 3   LIMITING RETRIEVAL PROPERTIES

How does reduction of dimension of "memory space" in the case $(B)$, $s \to s' = s-l$, affect retrieval properties of the system? They may vary considerably depending on $l$. In the most probable case $l = 1$ it is expected that there will be a slight decrease in storage capacity but the size of attraction basins will change negligibly. This is corroborated by calculating the stability parameter for each pattern $\mu$

$$\kappa_i^\mu = \xi_i^\mu \sum_{j \neq i} P'_{ij} \xi_j^\mu. \tag{5}$$

Let $S^{(m_1)}$ be the state vector with normalized projection on $\mathcal{L}$ given by $V = PS^{(m_1)}/|PS^{(m_1)}|$ such that

$$|PS^{(m_1)}| = \sqrt{\alpha N}, \quad V_i \sim N^{-1/2}, \quad \sum_{i=1}^{N} V_i \xi_i^\mu \sim 1.$$

Then the stability parameter (5) is estimated by

$$\kappa_i^\mu = \xi_i^\mu \sum_{j \neq i} (P_{ij} - V_i V_j) \xi_j^\mu = (1 - P_{ii}) - \left( V_i \xi_i^\mu \sum_{j=1}^{N} V_j \xi_j^\mu - V_i^2 \right) \approx 1 - P_{ii} + O(N^{-1/2}).$$

Since $P_{ii}$ has mean $\alpha$ and variance vanishing as $N \to \infty$, we thus conclude that the stability parameter only slightly differs from that calculated for the projector rule ($s = s'$) (Kanter & Sompolinsky, 1987).

On the other hand, in the situation $0 < s'/s \ll 1$ (the possible case $s' = 0$ is trivial) the system will be capable retrieving only a few nominated patterns which ones we cannot specify beforehand. As mentioned above, this case realizes with very small but finite probability.

The main effect of self-interactions $J_{ii}$ lies in substantial decrease in storage capacity (Kanter & Sompolinsky, 1987). This is relevant when considering the cases $(A)$ and $(B)$. In the case $(C)$ the system possesses an interesting dynamics exhibiting permanent walk over the state space. There are no fixed points at all. To show this, we write down the fixed point condition for arbitrary state $S$ : $S_i \sum_{j=1}^{N} J_{ij} S_j > 0$, $i = 1, \ldots, N$. By using the explicit expression for limiting matrix $\bar{J}_{ij}$ (3) and summing up over $i$'s, we get as a result $(\sum_j S_j \xi_j)^2 < 0$, what is impossible.

If self-interactions are excluded from local fields at the stage of network dynamics, it is then driven by the energy function of the form $H = -(2N)^{-1} \sum_{i \neq j} J_{ij} S_i S_j$. (Zero-temperature sequential dynamics either random or regular one is assumed.) In the rest of this section we examine dynamics of the network equiped with limiting synaptic matrix $(C)$ (3). We will show that in this limit the system lacks any associative memory. There are a single global maximum of $H$ given by $S_i = \text{sgn}(\xi_i)$ and exponentially many shallow minima concentrated close to the hyperplane orthogonal to $\xi$. Moreover it is turned out that all the metastable states are unstable against single spin flip only, whatever the realization of limiting vector $\xi$. Therefore after a spin flips the system can relax into a new nearby energy minimum. Through a sequence of steps each consisting of a single spin flip followed by relaxation one can, in principle, pass from one metastable state to the other one.

We will prove in what follows that *any given metastable state $S'$ one can pass to any other one $S$ through a sequence of steps each consisting of a single spin flip and subsequent relaxation to a some new metastable state*. Note that this general statement gives no indications concerning the order of spin flips when moving along a particular trajectory in the state space.

Now on we turn to the proof. Let us enumerate the spins in increasing order in absolute values of vector components $0 \leq |\xi_1| \leq \ldots \leq |\xi_N|$. The proof is carried out by induction on $j = 1, \ldots, N$ where $j$ is the maximal index for which $S_j^1 \neq S_j$.

For $j = 1$ the statement is evident. Assuming that it holds for $1, \ldots, j - 1$ ($2 \leq j \leq N$), let us prove it for $j$. One has $j = \max \{i : S_i^1 \neq S_i\}$. With flipping spin $j$ in the state $S^1$, we next allow relaxation by flipping spins $1, \ldots, j - 1$ only. The system finally reaches the state $S^2$ realizing conditional energy minimum under fixed $S_j, \ldots, S_N$.

Show that $S^2$ is true energy minimum. There are two possibilities:

(i) For some $i$, $1 \leq i \leq j - 1$, one has $\text{sgn}\left(\xi_i S_i^2\right) = \text{sgn}\left(\xi^T S^2\right)$. The fixed point condition for $S^2$ can be then written as

$$| \xi^T S^2 | \leq \min \left\{ |\xi_i| : \ 1 \leq i \leq j - 1, \ \text{sgn}(\xi_i S_i^2) = \text{sgn}(\xi^T S^2) \right\}.$$

¿From this, in view of increasing order of $|\xi_i|$'s, one gets immediately

$$| \xi^T S^2 | \leq \min \left\{ |\xi_i| : \ 1 \leq i \leq N, \ \text{sgn}(\xi_i S_i^2) = \text{sgn}(\xi^T S^2) \right\},$$

what implies $S^2$ is true energy minimum.

$(ii)$ $\operatorname{sgn}(\xi_i S_i^2) \neq \operatorname{sgn}(\xi^T S^2)$ for all $1 \leq i \leq j-1$.

If $\xi^T S^2 = 0$, the fixed point condition for $S^2$ is automatically satisfied. Otherwise, for $1 \leq i \leq j-1$ one has

$$\xi_i S_i^2 = -\operatorname{sgn}(\xi^T S^2)|\xi_i|,$$

and

$$\xi^T S^2 = -\operatorname{sgn}(\xi^T S^2) \sum_{i=1}^{j-1} |\xi_i| + \sum_{i=j}^{N} \xi_i S_i. \tag{6}$$

For the sake of definiteness, we set $\xi^T S > 0$. (The opposite case is treated analogously.) In this case $\xi^T S^2 > 0$, since otherwise, according to (6), it should be

$$0 \geq \xi^T S^2 = \sum_{i=1}^{j-1} |\xi_i| + \sum_{i=j}^{N} \xi_i S_i \geq \xi^T S,$$

what contradicts our setting.

One thus obtains

$$\xi^T S^2 = -\sum_{i=1}^{j-1} |\xi_i| + \sum_{i=j}^{N} \xi_i S_i \leq \xi^T S, \tag{7}$$

and using the fixed point condition for $S$ one gets

$$\xi^T S \leq \min\left\{|\xi_i| : \xi_i S_i > 0\right\} \leq \min\left\{|\xi_i| : j \leq i \leq N, \xi_i S_i > 0\right\}$$
$$= \min\left\{|\xi_i| : \xi_i S_i^2 > 0\right\}. \tag{8}$$

In the latter inequality of (8) one uses that $\xi_i S_i^2 < 0$, $1 \leq i \leq j-1$ and $S_i^2 = S_i$, $j \leq i \leq N$. Taking into account (7) and (8), as a result we come to the condition for $S^2$ to be true energy minimum

$$0 < \xi^T S^2 \leq \min\left\{|\xi_i| : \xi_i S_i^2 > 0\right\}.$$

According to inductive hypothesis, since $S_i^2 = S_i$, $j \leq i \leq N$, from the state $S^2$ one can pass to $S$, and therefore from $S'$ through $S^2$ to $S$. This proves the statement.

In general, metastable states may be grouped in clusters surrounded by high energy barriers. The meaning of proven statement resides in excluding the possibility of even such type a memory. Conversely, allowing a sequence of single spin flips (for instance, this can be done at finite temperatures) it is possible to walk through the whole set of metastable states.

## 4 CONCLUSION

In this paper we have begun studying on probabilities of different asymptotics of convergent unlearning algorithm considering the case of unbiased random patterns. We have shown also that failed unlearning results in total memory breakdown.

### References

Hopfield, J.J., Feinstein, D.I. & Palmer, R.G. (1983) "Unlearning" has a stabilizing effect in collective memories. *Nature* 304:158-159.

van Hemmen, J.L. & Klemmer, N. (1992) Unlearning and its relevance to REM sleep: Decorrelating correlated data. In J. G. Taylor et al (eds.), *Neural Network Dynamics*, pp. 30-43. London: Springer.

Wimbauer, U., Klemmer, N. & van Hemmen, J.L. (1994) Universality of unlearning. *Neural Networks* **7**:261-270.

Plakhov, A.Yu. & Semenov, S.A. (1994) Neural networks: iterative unlearning algorithm converging to the projector rule matrix. *J. Phys.I France* **4**:253-260.

Plakhov, A.Yu. (1995) private communication

Plakhov, A.Yu. & Semenov, S.A. (1995) preprint IPT.

Kanter, I. & Sompolinsky, H. (1987) Associative recall of memory without errors. *Phys. Rev. A* **35**:380-392.
